# Validity estimates for loopy Belief Propagation on binary real-world networks

**Joris Mooij**
Dept. of Biophysics, Inst. for Neuroscience, Radboud Univ. Nijmegen
6525 EZ Nijmegen, the Netherlands
`j.mooij@science.ru.nl`

**Hilbert J. Kappen**
Dept. of Biophysics, Inst. for Neuroscience, Radboud Univ. Nijmegen
6525 EZ Nijmegen, the Netherlands
`b.kappen@science.ru.nl`

## Abstract

We introduce a computationally efficient method to estimate the validity of the BP method as a function of graph topology, the connectivity strength, frustration and network size. We present numerical results that demonstrate the correctness of our estimates for the uniform random model and for a real-world network ("C. Elegans"). Although the method is restricted to pair-wise interactions, no local evidence (zero "biases") and binary variables, we believe that its predictions correctly capture the limitations of BP for inference and MAP estimation on arbitrary graphical models. Using this approach, we find that BP always performs better than MF. Especially for large networks with broad degree distributions (such as scale-free networks) BP turns out to significantly outperform MF.

## 1 Introduction

Loopy Belief Propagation (BP) [1] and its generalizations (such as the Cluster Variation Method [2]) are powerful methods for inference and optimization. As is well-known, BP is exact on trees, but also yields surprisingly good results for many other graphs that arise in real-world applications [3, 4]. On the other hand, for densely connected graphs with high interaction strengths the results can be quite bad or BP can simply fail to converge. Despite the fact that BP is often used in applications nowadays, a good theoretical understanding of its convergence properties and the quality of the approximation is still lacking (except for the very special case of graphs with a single loop [5]).

In this article we attempt to answer the question in what way the quality of the BP results depends on the topology of the underlying graph (looking at structural properties such as short cycles and large "hubs") and on the interaction potentials (i.e. strength and frustration). We do this for the special but interesting case of binary networks with symmetric pairwise potentials (i.e. Boltzmann machines) without local evidence. This has the practical

advantage that analytical calculations are feasible and furthermore we believe that adding local evidence will only serve to extend the domain of convergence, implying this to be the worst-case scenario. We compare the results with those of the variational mean-field (MF) method.

Real-world graphs are often far from uniformly random and possess structure such as clustering and power-law degree distributions [6]. Since we expect these structural features to arise in many applications of BP, we focus in this article on graphs modeling this kind of features. In particular, we consider Erdős-Rényi uniform random graphs [7], Barábasi-Albert "scale-free" graphs [8], and the neural network of a widely studied worm, the *Caenorhabditis elegans*.

This paper is organized as follows. In the next section we describe the class of graphical models under investigation and explain our method to efficiently estimate the validity of BP and MF. In section 3 we give a qualitative discussion of how the connectivity strength and frustration generally govern the model behavior and discuss the relevant regimes of the model parameters. We show for uniform random graphs that our validity estimates are in very good agreement with the real behavior of the BP algorithm. In section 4 we study the influence of graph topology. Thanks to the numerical efficiency of our estimation method we are able to study very large ($N \sim 10000$) networks, for which it would not be feasible to simply run BP and look what happens. We also try our method on the neural network of the worm C. Elegans and find almost perfect agreement of our predictions with observed BP behavior. We conclude that BP is always better than MF and that the difference is particularly striking for the case of large networks with broad degree distributions such as scale-free graphs.

## 2 Model, paramagnetic solution and stability analysis

Let $G = (V, B)$ be an undirected labelled graph without self-connections, defined by a set of nodes $V = \{1, \ldots, N\}$ and a set of links $B \subseteq \{(i,j) \,|\, 1 \leq i < j \leq N\}$. The *adjacency matrix* corresponding to $G$ is denoted $M$ and defined as follows: $M_{ij} := 1$ if $(ij) \in B$ or $(ji) \in B$ and 0 otherwise. We denote the set of neighbors of node $i \in V$ by $N_i := \{j \in V \,|\, (ij) \in B\}$ and its degree by $d_i := \#(N_i)$. We define the *average degree* $d := \frac{1}{N} \sum_{i \in V} d_i$ and the *maximum degree* $\Delta := \max_{i \in V} d_i$.

To each node $i$ we associate a binary random variable $x_i$ taking values in $\{-1, +1\}$. Let $W$ be a symmetric $N \times N$-matrix defining the strength of the links between the nodes. The probability distribution over configurations $\boldsymbol{x} = (x_1, \ldots, x_N)$ is given by

$$\mathbb{P}(\boldsymbol{x}) := \frac{1}{Z} \prod_{(ij) \in B} e^{W_{ij} x_i x_j} = \frac{1}{Z} \prod_{i,j \in V} e^{\frac{1}{2} M_{ij} W_{ij} x_i x_j} \tag{1}$$

with $Z$ a normalization constant. We will take the weight matrix $W$ to be random, with i.i.d. entries $\{W_{ij}\}_{1 \leq i < j \leq N}$ distributed according to the Gaussian law with mean $J_0$ and variance $J^2$.

For this model, instead of using the single-node and pair-wise beliefs $b_i(x_i)$ resp. $b_{ij}(x_i, x_j)$, it turns out to be more convenient to use the (equivalent) quantities $\boldsymbol{m} := \{m_i\}_{i \in V}$ and $\boldsymbol{\xi} := \{\xi_{ij}\}_{(ij) \in B}$, defined by:

$$m_i := b_i(+1) - b_i(-1);$$
$$\xi_{ij} := b_{ij}(+1, +1) - b_{ij}(+1, -1) - b_{ij}(-1, +1) + b_{ij}(-1, -1).$$

We will use these throughout this paper. We call the $m_i$ *magnetizations*; note that the expectation values $\mathbb{E}\, x_i$ vanish because of the symmetry in the probability distribution (1).

As is well-known [2, 9], fixed points of BP correspond to stationary points of the Bethe free energy, which is in this case given by

$$
F_{Be}(\boldsymbol{m}, \boldsymbol{\xi}) := - \sum_{(ij) \in B} W_{ij} \xi_{ij} + \sum_{i=1}^{N} (1 - d_i) \sum_{x_i = \pm 1} \eta \left( \frac{1 + m_i x_i}{2} \right)
$$
$$
+ \sum_{(ij) \in B} \sum_{x_i, x_j = \pm 1} \eta \left( \frac{1 + m_i x_i + m_j x_j + x_i x_j \xi_{ij}}{4} \right)
$$

with $\eta(x) := x \log x$. Note that with this parameterization all normalization and overlap constraints (i.e. $\sum_{x_j} b_{ij}(x_i, x_j) = b_i(x_i)$) are satisfied by construction [10]. We can minimize the Bethe free energy analytically by setting its derivatives to zero; one then immediately sees that a possible solution of the resulting equations is the *paramagnetic*[1] solution: $m_i = 0$ and $\xi_{ij} = \tanh W_{ij}$ (for $(ij) \in B$). For this solution to be a *minimum* (instead of a saddle point or maximum), the Hessian of $F_{Be}$ at that point should be positive-definite. This condition turns out to be equivalent to the following *Bethe stability matrix*

$$
(A_{Be})_{ij} := \delta_{ij} \left( 1 + \sum_{k \in N_i} \frac{\xi_{ik}^2}{1 - \xi_{ik}^2} \right) - M_{ij} \frac{\xi_{ij}}{1 - \xi_{ij}^2} \qquad \text{(with } \xi_{ij} = \tanh W_{ij}) \qquad (2)
$$

being positive-definite. Whether this is the case obviously depends on the values of the weights $W_{ij}$ and the adjacency matrix $M$. Since for zero weights ($W = 0$), the stability matrix is just the identity matrix, the paramagnetic solution is a minimum of the Bethe free energy for small values of the weights $W_{ij}$. The question of what "small" exactly means in terms of $J$ and $J_0$ and how this relates to the graph topology will be taken on in the next two sections.

First we discuss the situation for the mean-field variational method. The mean-field free energy $F_{MF}(\boldsymbol{m})$ only depends on $\boldsymbol{m}$; we can set its derivatives to zero, which again yields the paramagnetic solution $\boldsymbol{m} = 0$. The corresponding stability matrix (equal to the Hessian) is given by
$$
(A_{MF})_{ij} := \delta_{ij} - W_{ij} M_{ij}
$$
and should be positive-definite for the paramagnetic solution to be stable. One can prove [11] that $A_{Be}$ is positive-definite whenever $A_{MF}$ is positive-definite. Since the exact magnetizations are zero, we conclude that the Bethe approximation is better than the mean-field approximation for all possible choices of the weights $W$. As we will see later on, this difference can become quite large for large networks.

## 3 Weight dependence

The behavior of the graphical model depends critically on the parameters $J_0$ and $J$. Taking the graph topology to be uniformly random (see also subsection 4.1) we recover the model known in the statistical physics community as the Viana-Bray model [12], which has been thoroughly studied and is quite well-understood. In the limit $N \to \infty$, there are different relevant regimes ("phases") for the parameters $J$ and $J_0$ to be distinguished (cf. Fig. 1):

- The *paramagnetic phase*, where the magnetizations all vanish ($\boldsymbol{m} = 0$), valid for $J$ and $J_0$ both small.
- The *ferromagnetic phase*, where two configurations (characterized by all magnetizations being either positive or negative) each get half of the probability mass. This is the phase occurring for large $J_0$.

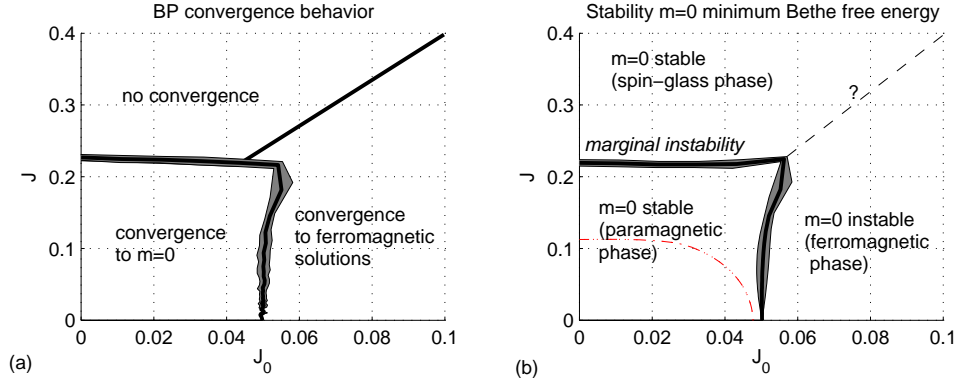

Figure 1: Empirical regime boundaries for the ER graph model with $N = 100$ and $d = 20$, averaged over three instances; expectation values are shown as thick black lines, standard-deviations are indicated by the gray areas. See the main text for additional explanation. The exact location of the boundary between the spin-glass and ferromagnetic phase in the right-hand plot (indicated by the dashed line) was not calculated. The red dash-dotted line shows the stability boundary for MF.

- The *spin-glass phase* where the probability mass is distributed over exponentially (in $N$) many different configurations. This phase occurs for frustrated weights, i.e. for large $J$.

Consider now the right-hand plot in Fig. 1. Here we have plotted the different regimes concerning the stability of the paramagnetic solution of the Bethe approximation.[2] We find that the $m = 0$ solution is indeed stable for $J$ and $J_0$ small and becomes unstable at some point when $J_0$ increases. This signals the paramagnetic-ferromagnetic phase transition. The location is in good agreement with the known phase boundary found for the $N \to \infty$ limit by advanced statistical physics methods as we show in more detail in [11]. For comparison we have also plotted the stability boundary for MF (the red dash-dotted line). Clearly, the mean-field approximation breaks down much earlier than the Bethe approximation and is unable to capture the phase transitions occurring for large connectivity strengths.

The boundary between the spin-glass phase and the paramagnetic phase is more subtle. What happens is that the Bethe stability matrix becomes *marginally stable* at some point when we increase $J$, i.e. the minimum eigenvalue of $A_{Be}$ approaches zero (in the limit $N \to \infty$). This means that the Bethe free energy becomes very flat at that point. If we go on increasing $J$, the $m = 0$ solution becomes stable again (in other words, the minimum eigenvalue of the stability matrix $A_{Be}$ becomes positive again). We interpret the marginal instability as signalling the onset of the spin-glass phase. Indeed it coincides with the known phase boundary for the Viana-Bray model [11, 12]. We observe a similar marginal instability for other graph topologies.

Now consider the left-hand plot, Fig. 1(a). It shows the convergence behavior of the BP algorithm, which was determined by running BP with a fixed number of maximum iterations and slight damping. The messages were initialized randomly. We find different regimes that are separated by the boundaries shown in the plot. For small $J$ and $J_0$, BP converges to $m = 0$. For $J_0$ large enough, BP converges to one of the two ferromagnetic solutions

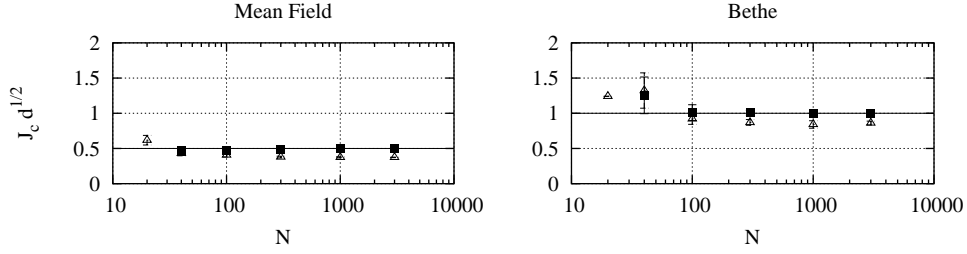

Figure 2: Critical values for Bethe and MF for different graph topologies (■: ER, △: BA) in the dense limit with $d = 0.1N$ as a function of network size. Note that the $y$-axis is rescaled by $\sqrt{d}$.

(which one is determined by the random initial conditions). For large $J$, BP does not converge within 1000 iterations, indicating a complex probability distribution. The boundaries coincide within statistical precision with those in the right-hand plot which were obtained by the stability analysis.

The computation time necessary for producing a plot such as Fig. 1(a), showing the convergence behavior of BP, quickly increases with increasing $N$. The computation time needed for the stability analysis (Fig. 1(b)), which amounts to calculating the minimal eigenvalue of the $N \times N$ stability matrix, is much less, allowing us to investigate the behavior of BP for large networks.

## 4   Graph topology

In this section we will concentrate on the frustrated case, more precisely on the case $J_0 = 0$ (i.e. the $y$-axis in the regime diagrams) and study the location of the Bethe marginal instability and of the MF instability for various graph topologies as a function of network size $N$ and average degree $d$. We will denote by $J_c^{Be}$ the critical value of $J$ at which the Bethe paramagnetic solution becomes marginally unstable and we will refer to this as the *Bethe critical value*. The critical value of $J$ where the MF solution becomes unstable will be denoted as $J_c^{MF}$ and referred to as the *MF critical value*.

In studying the influence of graph topology for large networks, we have to distinguish two cases, which we call the *dense* and *sparse* limits. In the dense limit, we let $N \to \infty$ and scale the average degree as $d = cN$ for some fixed constant $c$. In this limit, we find that the influence of the graph topology is almost negligible. For all graph topologies that we have considered, we find the following asymptotic behavior for the critical values:

$$J_c^{Be} \propto \frac{1}{\sqrt{d}}, \qquad J_c^{MF} \propto \frac{1}{2\sqrt{d}}$$

The constant of proportionality is approximately 1. These results are illustrated in Fig. 2 for two different graph topologies that will be discussed in more detail below.

In the sparse limit, we let $N \to \infty$ but keep $d$ fixed. In that case the resulting critical values show significant dependence on the graph topology as we will see.

### 4.1   Uniform random graphs (ER)

The first and most elementary random graph model we will consider was introduced and studied by Erdős and Rényi [7]. The ensemble, which we denote as $ER(N, p)$, consists of

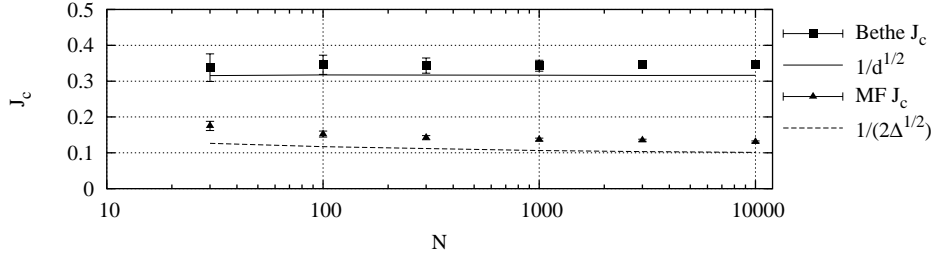

Figure 3: Critical values for Bethe and MF for Erdős-Rényi uniform random graphs with average degree $d = 10$.

the graphs with $N$ nodes; links are added between each pair of nodes independently with probability $p$. The resulting graphs have a degree distribution that is approximately Poisson for large $N$ and the expected average degree is $\mathbb{E} d = p(N-1)$. As was mentioned before, the resulting graphical model is known in the statistical physics literature as the Viana-Bray model (with zero "external field").

Fig. 3 shows the results for the sparse limit, where $p$ is chosen such that the expected average degree is fixed to $d = 10$. The Bethe critical value $J_c^{Be}$ appears to be independent of network size and is slightly larger than $1/\sqrt{d}$. The MF critical value $J_c^{MF}$ does depend on network size (it looks to be proportional to $1/\sqrt{\Delta}$ instead of $1/\sqrt{d}$); in fact it can be proven that it converges very slowly to 0 as $N \rightarrow \infty$ [11], implying that the MF approximation breaks down for very large ER networks in the sparse limit. Although this is an interesting result, one could say that for all practical purposes the MF critical value $J_c^{MF}$ is nearly independent of network size $N$ for uniform random graphs.

## 4.2 Scale-free graphs (BA)

A phenomenon often observed in real-world networks is that the degree distribution behaves like a power-law, i.e. the number of nodes with degree $\delta$ is proportional to $\delta^{-\alpha}$ for some $\alpha > 0$. These graphs are also known as "scale-free" graphs. The first random graph model exhibiting this behavior is from Barabási and Albert [8].

We will consider a slightly different model, which we will denote by $BA(N, m)$. It is defined as a stochastic process, yielding graphs with more and more nodes as time goes on. At $t = 0$ one starts with the graph consisting of $m$ nodes and no links. At each time step, one node is added; it is connected with $m$ different already existing nodes, attaching preferably to nodes with higher degree ("rich get richer"). More specifically, we take the probability to connect to a node of degree $\delta$ to be proportional to $\delta + 1$. The degree distribution turns out to have a power-law dependence for $N \rightarrow \infty$ with exponent $\alpha = 3$. In Fig. 4 we illustrate some BA graphs. The difference between the maximum degree $\Delta$ and the average degree $d$ is rather large: whereas the average degree $d$ converges to $2m$, the maximum degree $\Delta$ is known to scale as $\sqrt{N}$.

Fig. 5 shows the results of the stability analysis for BA graphs with average degree $d = 10$. Note that the $y$-axis is rescaled by $\sqrt{\Delta}$ to show that the MF critical value $J_c^{MF}$ is proportional to $1/\sqrt{\Delta}$. The Bethe critical values are seen to have a scaling behavior that lies somewhere between $1/\sqrt{d}$ and $1/\sqrt{\Delta}$. Compared to the situation for uniform ER graphs, BP now even more significantly outperforms MF. The relatively low sensitivity to the maximum degree $\Delta$ that BP exhibits here can be understood intuitively since BA graphs resemble forests of sparsely interconnected stars of high degree, on which BP is exact.

### 4.3 C. Elegans

We have also applied our stability analysis on the neural network of the worm C. Elegans, that is publicly available on `http://elegans.swmed.edu/`. This graph has $N = 202$ and $d = 19.4$. We have calculated the ferromagnetic ($J = 0$) transition and spin-glass ($J_0 = 0$) transition. We also calculated the critical value of $J$ where BP stops converging, and the value of $J$ where BP does not find the paramagnetic solution anymore. The results are shown in Table 1. Note the very good agreement for the Bethe critical value and the critical $J$ where BP stops finding the $m = 0$ solution. These results show the accuracy of our method of estimating BP validity on real-world networks.

Table 1: Critical values and BP boundaries for C. Elegans network.

|  | Spin-glass | Ferromagnetic |
|---|---|---|
| MF critical value | $0.0927 \pm 0.0023$ | $0.0387$ |
| Bethe critical value | $0.197 \pm 0.016$ | $0.0406$ |
| BP $m = 0$ boundary | $0.194 \pm 0.014$ | $0.0400$ |
| BP convergence boundary | $0.209 \pm 0.027$ | $> 1$ |

## 5 Conclusions

We have introduced a computationally efficient method to estimate the validity of BP as a function of graph topology, the connectivity strength, frustration and network size. Using this approach, we have found that:

- for any graph, the Bethe approximation is valid for a larger set of connectivity strengths $W_{ij}$ than the mean-field approximation;

- for uniform random graphs, the quality of both the MF approximation and the Bethe approximation is determined by the average degree of the network ($J_c \propto 1/\sqrt{d}$ for the spin-glass transition) and is nearly independent of network size;

- for scale-free networks the validity of the MF approximation scales very poorly with network size due to the increase of the maximal degree ("rich get richer"). In contrast, the validity of the BP approximation scales very well with network size. This is in agreement with our intuition that these networks resemble a forest of high degree stars ("hubs") that are sparsely interconnected and the fact that BP is exact on stars.

- In the limit in which the graph size $N \to \infty$ and the average degree $d$ scales proportional to $N$, the influence of the graph-topological details on the location of the spin-glass transition (at $J \propto 1/\sqrt{d}$) diminishes and becomes largely irrelevant.

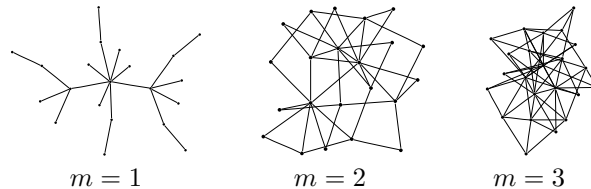

$m = 1$      $m = 2$      $m = 3$

Figure 4: Barábasi-Albert graphs for $N = 20$.

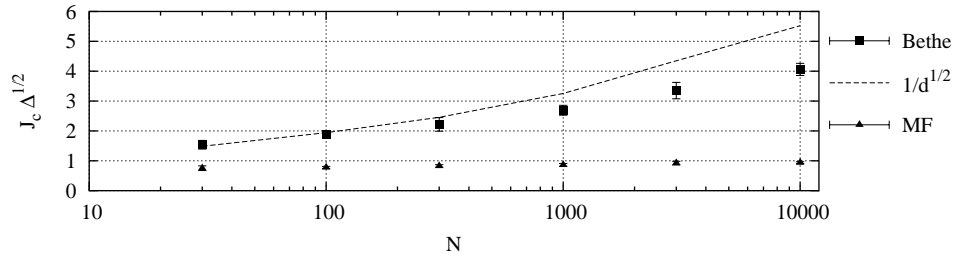

Figure 5: Critical values for Bethe and MF for BA scale-free random graphs with average degree $d = 10$. Note that the $y$-axis is rescaled by $\sqrt{\Delta}$.

**Acknowledgments**

The research reported here is part of the Interactive Collaborative Information Systems (ICIS) project, supported by the Dutch Ministry of Economic Affairs, grant BSIK03024.

## Footnotes

[1] Throughout this article, we will use terminology from statistical physics if there is no good corresponding terminology in the field of machine learning available.

[2]Although in Fig. 1 we show only one particular graph topology, the general appearance of these plots does not differ much for other graph topologies, especially for large $N$. The scale of the plots mostly depends on the network size $N$ and the average degree $d$ as we will show in the next section.

## References

[1] J. Pearl. *Probabilistic Reasoning in Intelligent systems: Networks of Plausible Inference*. Morgan Kaufmann, San Francisco, CA, 1988.

[2] J. Yedidia, W. Freeman, and Y. Weiss. Generalized belief propagation. In *Advances in Neural Information Processing Systems*, volume 13, pages 689–695, 2001.

[3] K. Murphy, Y. Weiss, and M. Jordan. Loopy belief propagation for approximate inference: an empirical study. In *Proc. of the Conf. on Uncertainty in AI*, pages 467–475, 1999.

[4] B. Frey and D. MacKay. A revolution: Belief propagation in graphs with cycles. In *Advances in Neural Information Processing Systems*, volume 10, pages 479–485, 1997.

[5] Y. Weiss. Correctness of local probability propagation in graphical models with loops. *Neur. Comp.*, 12:1–41, 2000.

[6] R. Albert and A.-L. Barabási. Statistical mechanics of complex networks. *Rev. Mod. Phys.*, 74:47–97, 2002.

[7] P. Erdős and A. Rényi. On random graphs i. *Publ. Math. Debrecen*, 6:290–291, 1959.

[8] A.-L. Barabási and R. Albert. Emergence of scaling in random networks. *Science*, 286:509–512, 1999.

[9] T. Heskes. Stable fixed points of loopy belief propagation are local minima of the bethe free energy. In *Advances in Neural Information Processing Systems*, volume 15, pages 343–350, 2003.

[10] M. Welling and Y.W. Teh. Belief optimization for binary networks: a stable alternative to loopy belief propagation. In *Proc. of the Conf. on Uncertainty in AI*, volume 17, 2001.

[11] J.M. Mooij and H.J. Kappen. Spin-glass phase transitions on real-world graphs. *preprint*, cond-mat:0408378, 2004.

[12] L. Viana and A. Bray. Phase diagrams for dilute spin glasses. *J. Phys. C: Solid State Phys.*, 18:3037–3051, 1985.
